# Multi-Task Feature Learning

**Andreas Argyriou**
Department of Computer Science
University College London
Gower Street, London WC1E 6BT, UK
*a.argyriou@cs.ucl.ac.uk*

**Theodoros Evgeniou**
Technology Management and Decision Sciences,
INSEAD,
Bd de Constance, Fontainebleau 77300, France
*theodoros.evgeniou@insead.edu*

**Massimiliano Pontil**
Department of Computer Science
University College London
Gower Street, London WC1E 6BT, UK
*m.pontil@cs.ucl.ac.uk*

## Abstract

We present a method for learning a low-dimensional representation which is shared across a set of multiple related tasks. The method builds upon the well-known 1-norm regularization problem using a new regularizer which controls the number of learned features common for all the tasks. We show that this problem is equivalent to a convex optimization problem and develop an iterative algorithm for solving it. The algorithm has a simple interpretation: it alternately performs a supervised and an unsupervised step, where in the latter step we learn common-across-tasks representations and in the former step we learn task-specific functions using these representations. We report experiments on a simulated and a real data set which demonstrate that the proposed method dramatically improves the performance relative to learning each task independently. Our algorithm can also be used, as a special case, to simply select – not learn – a few common features across the tasks.

## 1 Introduction

Learning multiple related tasks simultaneously has been empirically [2, 3, 8, 9, 12, 18, 19, 20] as well as theoretically [2, 4, 5] shown to often significantly improve performance relative to learning each task independently. This is the case, for example, when only a few data per task are available, so that there is an advantage in "pooling" together data across many related tasks.

Tasks can be related in various ways. For example, task relatedness has been modeled through assuming that all functions learned are close to each other in some norm [3, 8, 15, 19]. This may be the case for functions capturing preferences in users' modeling problems [9, 13]. Tasks may also be related in that they all share a common underlying representation [4, 5, 6]. For example, in object recognition, it is well known that the human visual system is organized in a way that all objects[1] are represented – at the earlier stages of the visual system – using a common set of features learned, e.g. local filters similar to wavelets [16]. In modeling users' preferences/choices, it may also be the case that people make product choices (e.g. of books, music CDs, etc.) using a common set of features describing these products.

In this paper, we explore the latter type of task relatedness, that is, we wish to learn a low-dimensional representation which is shared across multiple related tasks. Inspired by the fact that the well known $1-$norm regularization problem provides such a sparse representation for the single

task case, in Section 2 we generalize this formulation to the multiple task case. Our method learns a few features common across the tasks by regularizing within the tasks while keeping them coupled to each other. Moreover, the method can be used, as a special case, to select (not learn) a few features from a prescribed set. Since the extended problem is nonconvex, we develop an equivalent convex optimization problem in Section 3 and present an algorithm for solving it in Section 4. A similar algorithm was investigated in [9] from the perspective of conjoint analysis. Here we provide a theoretical justification of the algorithm in connection with 1-norm regularization.

The learning algorithm simultaneously learns *both* the features and the task functions through two alternating steps. The first step consists of independently learning the parameters of the tasks' regression or classification functions. The second step consists of learning, in an unsupervised way, a low-dimensional representation for these task parameters, which we show to be equivalent to learning common features across the tasks. The number of common features learned is controlled, as we empirically show, by the regularization parameter, much like sparsity is controlled in the case of single-task 1-norm regularization.

In Section 5, we report experiments on a simulated and a real data set which demonstrate that the proposed method learns a few common features across the tasks while also improving the performance relative to learning each task independently. Finally, in Section 6 we briefly compare our approach with other related multi-task learning methods and draw our conclusions.

## 2    Learning sparse multi-task representations

We begin by introducing our notation. We let $\mathbb{R}$ be the set of real numbers and $\mathbb{R}_+$ ($\mathbb{R}_{++}$) the subset of non-negative (positive) ones. Let $T$ be the number of tasks and define $\mathbb{N}_T := \{1, \ldots, T\}$. For each task $t \in \mathbb{N}_T$, we are given $m$ input/output examples $(x_{t1}, y_{t1}), \ldots (x_{tm}, y_{tm}) \in \mathbb{R}^d \times \mathbb{R}$. Based on this data, we wish to estimate $T$ functions $f_t : \mathbb{R}^d \to \mathbb{R}$, $t \in \mathbb{N}_T$, which approximate well the data and are statistically predictive, see e.g. [11].

If $w, u \in \mathbb{R}^d$, we define $\langle w, u \rangle := \sum_{i=1}^d w_i u_i$, the standard inner product in $\mathbb{R}^d$. For every $p \geq 1$, we define the $p$-norm of vector $w$ as $\|w\|_p := (\sum_{i=1}^d |w_i|^p)^{\frac{1}{p}}$. If $A$ is a $d \times T$ matrix we denote by $a^i \in \mathbb{R}^T$ and $a_j \in \mathbb{R}^d$ the $i$-th row and the $j$-th column of $A$ respectively. For every $r, p \geq 1$ we define the $(r, p)$-norm of $A$ as $\|A\|_{r,p} := \left( \sum_{i=1}^d \|a^i\|_r^p \right)^{\frac{1}{p}}$.

We denote by $\mathbf{S}^d$ the set of $d \times d$ real symmetric matrices and by $\mathbf{S}_+^d$ the subset of positive semidefinite ones. If $D$ is a $d \times d$ matrix, we define $\text{trace}(D) := \sum_{i=1}^d D_{ii}$. If $X$ is a $p \times q$ real matrix, $\text{range}(X)$ denotes the set $\{x \in \mathbb{R}^p : x = Xz, \text{ for some } z \in \mathbb{R}^q\}$. We let $\mathbf{O}^d$ be the set of $d \times d$ orthogonal matrices. Finally, $D^+$ denotes the pseudoinverse of a matrix $D$.

### 2.1    Problem formulation

The underlying assumption in this paper is that the functions $f_t$ are related so that they *all share a small set of features*. Formally, our hypothesis is that the functions $f_t$ can be represented as

$$f_t(x) = \sum_{i=1}^d a_{it} h_i(x), \qquad t \in \mathbb{N}_T, \tag{2.1}$$

where $h_i : \mathbb{R}^d \to \mathbb{R}$ are the features and $a_{it} \in \mathbb{R}$ are the regression parameters. Our main assumption is that all the features but a few have zero coefficients across *all* the tasks.

For simplicity, we focus on linear features, that is, $h_i(x) = \langle u_i, x \rangle$, where $u_i \in \mathbb{R}^d$. In addition, we assume that the vectors $u_i$ are orthonormal. Thus, if $U$ denotes the $d \times d$ matrix with columns the vectors $u_i$, then $U \in \mathbf{O}^d$. The functions $f_t$ are linear as well, that is $f_t(x) = \langle w_t, x \rangle$, where $w_t = \sum_i a_{it} u_i$. Extensions to nonlinear functions may be done, for example, by using kernels along the lines in [8, 15]. Since this is not central in the present paper we postpone its discussion to a future occasion.

Let us denote by $W$ the $d \times T$ matrix whose columns are the vectors $w_t$ and by $A$ the $d \times T$ matrix with entries $a_{it}$. We then have that $W = UA$. Our assumption that the tasks share a "small" set

of features means that the matrix $A$ has "many" rows which are identically equal to zero and, so, the corresponding features (columns of matrix $U$) will not be used to represent the task parameters (columns of matrix $W$). In other words, matrix $W$ is a low rank matrix. We note that the problem of learning a low-rank matrix factorization which approximates a given partially observed target matrix has been considered in [1], [17] and references therein. We briefly discuss its connection to our current work in Section 4.

In the following, we describe our approach to computing the feature vectors $u_i$ and the parameters $a_{it}$. We first consider the case that there is only one task (say task $t$) and the features $u_i$ are fixed. To learn the parameter vector $a_t \in \mathbb{R}^d$ from data $\{(x_{ti}, y_{ti})\}_{i=1}^m$ we would like to minimize the empirical error $\sum_{i=1}^m L(y_{ti}, \langle a_t, U^\top x_{ti} \rangle)$ subject to an upper bound on the number of nonzero components of $a_t$, where $L : \mathbb{R} \times \mathbb{R} \to \mathbb{R}_+$ is a prescribed loss function which we assume to be convex in the second argument. This problem is intractable and is often relaxed by requiring an upper bound on the 1-norm of $a_t$. That is, we consider the problem $\min \left\{ \sum_{i=1}^m L(y_{ti}, \langle a_t, U^\top x_{ti} \rangle) : \|a_t\|_1^2 \leq \alpha^2 \right\}$, or equivalently the unconstrained problem

$$\min \left\{ \sum_{i=1}^m L(y_{ti}, \langle a_t, U^\top x_{ti} \rangle) + \gamma \|a_t\|_1^2 \; : \; a_t \in \mathbb{R}^d \right\}, \tag{2.2}$$

where $\gamma > 0$ is the regularization parameter. It is well known that using the 1-norm leads to sparse solutions, that is, many components of the learned vector $a_t$ are zero, see [7] and references therein. Moreover, the number of nonzero components of a solution to problem (2.2) is "typically" a non-increasing function of $\gamma$ [14].

We now generalize problem (2.2) to the multi-task case. For this purpose, we introduce the regularization error function

$$\mathcal{E}(A, U) = \sum_{t=1}^T \sum_{i=1}^m L(y_{ti}, \langle a_t, U^\top x_{ti} \rangle) + \gamma \|A\|_{2,1}^2. \tag{2.3}$$

The first term in (2.3) is the average of the empirical error across the tasks while the second one is a regularization term which penalizes the $(2, 1)$-norm of the matrix $A$. It is obtained by first computing the 2-norm of the (across the tasks) rows $a^i$ (corresponding to feature $i$) of matrix $A$ and then the 1-norm of the vector $b(A) = (\|a^1\|_2, \dots, \|a^d\|_2)$. This norm combines the tasks and ensures that common features will be selected across them.

Indeed, if the features $U$ are prescribed and $\hat{A}$ minimizes the function $\mathcal{E}$ over $A$, the number of nonzero components of the vector $b(\hat{A})$ will typically be non-increasing with $\gamma$ like in the case of 1-norm single-task regularization. Moreover, the components of the vector $b(\hat{A})$ indicate how important each feature is and favor uniformity across the tasks for each feature.

Since we do not simply want to *select* the features but also *learn* them, we further minimize the function $\mathcal{E}$ over $U$, that is, we consider the optimization problem

$$\min \left\{ \mathcal{E}(A, U) : U \in \mathbf{O}^d, A \in \mathbb{R}^{d \times T} \right\}. \tag{2.4}$$

This method learns a low-dimensional representation which is shared across the tasks. As in the single-task case, the number of features will be typically non-increasing with the regularization parameter – we shall present experimental evidence of this fact in Section 5 (see Figure 1 therein).

We note that when the matrix $U$ is not learned and we set $U = I_{d \times d}$, problem (2.4) computes a common set of variables across the tasks. That is, we have the following convex optimization problem

$$\min \left\{ \sum_{t=1}^T \sum_{i=1}^m L(y_{ti}, \langle a_t, x_{ti} \rangle) + \gamma \|A\|_{2,1}^2 \; : \; A \in \mathbb{R}^{d \times T} \right\}. \tag{2.5}$$

We shall return to problem (2.5) in Section 4 where we present an algorithm for solving it.

## 3   Equivalent convex optimization formulation

Solving problem (2.4) is a challenging task for two main reasons. First, it is a non-convex problem, although it is separately convex in each of the variables $A$ and $U$. Second, the norm $\|A\|_{2,1}$ is nonsmooth which makes it more difficult to optimize.

A main result in this paper is that problem (2.4) can be transformed into an equivalent convex problem. To this end, for every $W \in \mathbb{R}^{d \times T}$ and $D \in \mathbf{S}_+^d$, we define the function

$$\mathcal{R}(W, D) = \sum_{t=1}^{T} \sum_{i=1}^{m} L(y_{ti}, \langle w_t, x_{ti} \rangle) + \gamma \sum_{t=1}^{T} \langle w_t, D^+ w_t \rangle. \qquad (3.1)$$

**Theorem 3.1.** *Problem (2.4) is equivalent to the problem*

$$\min \left\{ \mathcal{R}(W, D) \ : \ W \in \mathbb{R}^{d \times T}, \ D \in \mathbf{S}_+^d, \ \mathrm{trace}(D) \leq 1, \ \mathrm{range}(W) \subseteq \mathrm{range}(D) \right\}. \qquad (3.2)$$

*That is, $(\hat{A}, \hat{U})$ is an optimal solution for (2.4) if and only if $(\hat{W}, \hat{D}) = (\hat{U}\hat{A}, \hat{U}\mathrm{Diag}(\hat{\lambda})\hat{U}^\top)$ is an optimal solution for (3.2), where*

$$\hat{\lambda}_i := \frac{\|\hat{a}^i\|_2}{\|\hat{A}\|_{2,1}}. \qquad (3.3)$$

**Proof.** Let $W = UA$ and $D = U\mathrm{Diag}(\frac{\|a^i\|_2}{\|A\|_{2,1}})U^\top$. Then $\|a^i\|_2 = \|W^\top u_i\|_2$ and hence

$$\sum_{t=1}^{T} \langle w_t, D^+ w_t \rangle = \mathrm{trace}(W^\top D^+ W) = \|A\|_{2,1} \, \mathrm{trace}(W^\top U \mathrm{Diag}(\|W^\top u_i\|_2)^+ U^\top W) =$$

$$\|A\|_{2,1} \, \mathrm{trace}\Big(\sum_{i=1}^{d}(\|W^\top u_i\|_2)^+ \, W^\top u_i u_i^\top W\Big) = \|A\|_{2,1} \sum_{i=1}^{d} \|W^\top u_i\|_2 = \|A\|_{2,1}^2 \, .$$

Therefore, $\min_{W,D} \mathcal{R}(W, D) \leq \min_{A,U} \mathcal{E}(A, U)$. Conversely, let $D = U\mathrm{Diag}(\lambda)U^\top$. Then

$$\sum_{t=1}^{T} \langle w_t, D^+ w_t \rangle = \mathrm{trace}(W^\top U \mathrm{Diag}(\lambda_i^+)U^\top W) = \mathrm{trace}(\mathrm{Diag}(\lambda_i^+)AA^\top) \geq \|A\|_{2,1}^2 \, ,$$

by Lemma 4.2. Note that the range constraint ensures that $W$ is a multiple of the submatrix of $U$ which corresponds to the nonzero eigenvalues of $D$, and hence if $\lambda_i = 0$ then $a^i = 0$ as well. Therefore, $\min_{A,U} \mathcal{E}(A, U) \leq \min_{W,D} \mathcal{R}(W, D)$. $\qquad \square$

In problem (3.2) we have constrained the trace of $D$, otherwise the optimal solution would be to simply set $D = \infty$ and only minimize the empirical error term in (3.1). Similarly, we have imposed the range constraint to ensure that the penalty term is bounded below and away from zero. Indeed, without this constraint, it may be possible that $DW = 0$ when $W$ does not have full rank, in which case there is a matrix $D$ for which $\sum_{t=1}^{T} \langle w_t, D^+ w_t \rangle = \mathrm{trace}(W^\top D^+ W) = 0$.

We note that the rank of matrix $D$ indicates how many common relevant features the tasks share. Indeed, it is clear from equation (3.3) that the rank of matrix $D$ equals the number of nonzero rows of matrix $A$.

We now show that the function $\mathcal{R}$ in equation (3.1) is jointly convex in $W$ and $D$. For this purpose, we define the function $f(w, D) = w^\top D^+ w$, if $D \in \mathbf{S}_+^d$ and $w \in \mathrm{range}(D)$, and $f(w, D) = +\infty$ otherwise. Clearly, $\mathcal{R}$ is convex provided $f$ is convex. The latter is true since a direct computation expresses $f$ as the supremum of a family of convex functions, namely we have that $f(w, D) = \sup\{w^\top v + \mathrm{trace}(ED) \ : \ E \in \mathbf{S}^d, v \in \mathbb{R}^d, 4E + vv^\top \preceq 0\}$.

## 4 Learning algorithm

We solve problem (3.2) by alternately minimizing the function $\mathcal{R}$ with respect to $D$ and the $w_t$ (recall that $w_t$ is the $t$-th column of matrix $W$).

When we keep $D$ fixed, the minimization over $w_t$ simply consists of learning the parameters $w_t$ independently by a regularization method, for example by an SVM or ridge regression type method[2]. For a fixed value of the vectors $w_t$, we learn $D$ by simply solving the minimization problem

$$\min \left\{ \sum_{t=1}^{T} \langle w_t, D^+ w_t \rangle \ : \ D \in \mathbf{S}_+^d, \ \mathrm{trace}(D) \leq 1, \ \mathrm{range}(W) \subseteq \mathrm{range}(D) \right\}. \qquad (4.1)$$

The following theorem characterizes the optimal solution of problem (4.1).

**Algorithm 1** (*Multi-Task Feature Learning*)

---

**Input:** training sets $\{(x_{ti}, y_{ti})\}_{i=1}^{m}, t \in \mathbb{N}_T$
**Parameters:** regularization parameter $\gamma$
**Output:** $d \times d$ matrix $D$, $d \times T$ regression matrix $W = [w_1, \ldots, w_T]$
**Initialization:** set $D = \frac{I_{d \times d}}{d}$
**while** convergence condition is not true **do**
    **for** $t = 1, \ldots, T$ **do**
        compute $w_t = \operatorname{argmin} \left\{ \sum_{i=1}^{m} L(y_{ti}, \langle w, x_{ti} \rangle) + \gamma \langle w, D^+ w \rangle : w \in \mathbb{R}^d, \ w \in \operatorname{range}(D) \right\}$
    **end for**
    set $D = \frac{(WW^\top)^{\frac{1}{2}}}{\operatorname{trace}(WW^\top)^{\frac{1}{2}}}$
**end while**

---

**Theorem 4.1.** *Let $C = WW^\top$. The optimal solution of problem (4.1) is*

$$D = \frac{C^{\frac{1}{2}}}{\operatorname{trace} C^{\frac{1}{2}}} \tag{4.2}$$

*and the optimal value equals $(\operatorname{trace} C^{\frac{1}{2}})^2$.*

We first introduce the following lemma which is useful in our analysis.

**Lemma 4.2.** *For any $b = (b_1, \ldots, b_d) \in \mathbb{R}^d$, we have that*

$$\inf \left\{ \sum_{i=1}^{d} \frac{b_i^2}{\lambda_i} \ : \ \lambda_i > 0, \ \sum_{i=1}^{d} \lambda_i \leq 1 \right\} = \|b\|_1^2 \tag{4.3}$$

*and any minimizing sequence converges to $\hat{\lambda}_i = \frac{|b_i|}{\|b\|_1}, i \in \mathbb{N}_d$.*

**Proof.** From the Cauchy-Schwarz inequality we have that $\|b\|_1 = \sum_{b_i \neq 0} \lambda_i^{\frac{1}{2}} \lambda_i^{-\frac{1}{2}} |b_i| \leq (\sum_{b_i \neq 0} \lambda_i)^{\frac{1}{2}} (\sum_{b_i \neq 0} \lambda_i^{-1} b_i^2)^{\frac{1}{2}} \leq (\sum_{i=1}^{d} \lambda_i^{-1} b_i^2)^{\frac{1}{2}}$. Convergence to the infimum is obtained when $\sum_{i=1}^{d} \lambda_i \to 1$ and $\frac{|b_i|}{\lambda_i} - \frac{|b_j|}{\lambda_j} \to 0$ for all $i, j \in \mathbb{N}_d$ such that $b_i, b_j \neq 0$. Hence $\lambda_i \to \frac{|b_i|}{\|b\|_1}$. The infimum is attained when $b_i \neq 0$ for all $i \in \mathbb{N}_d$. $\square$

**Proof of Theorem 4.1.** We write $D = U\operatorname{Diag}(\lambda)U^\top$, with $U \in \mathbf{O}^d$ and $\lambda \in \mathbb{R}_+^d$. We first minimize over $\lambda$. For this purpose, we use Lemma 4.2 to obtain that

$$\inf\{\operatorname{trace}(W^\top U \operatorname{Diag}(\lambda)^{-1} U^\top W) : \lambda \in \mathbb{R}_{++}^d, \sum_{i=1}^{d} \lambda_i \leq 1\} = \|U^\top W\|_{2,1}^2 = \left( \sum_{i=1}^{d} \|W^\top u_i\|_2 \right)^2.$$

Next we show that

$$\min\{\|U^\top W\|_{2,1}^2 : U \in \mathbf{O}_d\} = (\operatorname{trace} C^{\frac{1}{2}})^2$$

and a minimizing $U$ is a system of eigenvectors of $C$. To see this, note that

$$\begin{aligned}
\operatorname{trace}(WW^\top u_i u_i^\top) &= \operatorname{trace}(C^{\frac{1}{2}} u_i u_i^\top u_i u_i^\top C^{\frac{1}{2}}) \operatorname{trace}(u_i u_i^\top u_i u_i^\top) \\
&\geq (\operatorname{trace}(C^{\frac{1}{2}} u_i u_i^\top u_i u_i^\top))^2 = \operatorname{trace}(C^{\frac{1}{2}} u_i u_i^\top) = u_i^\top C^{\frac{1}{2}} u_i
\end{aligned}$$

since $u_i u_i^\top u_i u_i^\top = u_i u_i^\top$. The equality is verified if and only if $C^{\frac{1}{2}} u_i u_i^\top = a u_i u_i^\top$ which implies that $C^{\frac{1}{2}} u_i = a u_i$, that is, if $u_i$ is an eigenvector of $C$. The optimal $a$ is $\operatorname{trace}(C^{\frac{1}{2}})$. $\square$

The expression $\operatorname{trace}(WW^\top)^{\frac{1}{2}}$ in (4.2) is simply the sum of the singular values of $W$ and is sometimes called the *trace norm*. As shown in [10], the trace norm is the convex envelope of $\operatorname{rank}(W)$ in the unit ball, which gives another interpretation of the relationship between the rank and $\gamma$ in our experiments. Using the trace norm, problem (3.2) becomes a regularization problem which depends only on $W$.

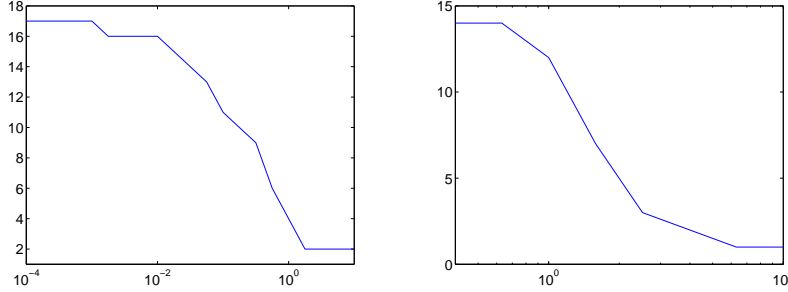

Figure 1: Number of features learned versus the regularization parameter $\gamma$ (see text for description).

However, since the trace norm is nonsmooth, we have opted for the above alternating minimization strategy which is simple to implement and has a natural interpretation. Indeed, Algorithm 1 alternately performs a supervised and an unsupervised step, where in the latter step we learn common representations across the tasks and in the former step we learn task-specific functions using these representations.

We conclude this section by noting that when matrix $D$ in problem (3.2) is additionally constrained to be diagonal, problem (3.2) reduces to problem (2.5). Formally, we have the following corollary.

**Corollary 4.3.** *Problem (2.5) is equivalent to the problem*

$$\min\left\{\mathcal{R}(W, \operatorname{Diag}(\lambda)) \ : \ W \in \mathbb{R}^{d \times T}, \ \lambda \in \mathbb{R}_+^d, \ \sum_{i=1}^{d} \lambda_i \leq 1, \ \lambda_i \neq 0 \ when \ w^i \neq 0\right\} \quad (4.4)$$

*and the optimal $\lambda$ is given by*

$$\lambda_i = \frac{\|w^i\|_2}{\|W\|_{2,1}}, \qquad i \in \mathbb{N}_d. \quad (4.5)$$

Using this corollary we can make a simple modification to Algorithm 1 in order to use it for variable selection. That is, we modify the computation of the matrix $D$ (penultimate line in Algorithm 1) as $D = Diag(\lambda)$, where the vector $\lambda = (\lambda_1, \ldots, \lambda_d)$ is computed using equation (4.5).

## 5   Experiments

In this section, we present experiments on a synthetic and a real data set. In all of our experiments, we used the square loss function and automatically tuned the regularization parameter $\gamma$ with leave-one-out cross validation.

**Synthetic Experiments.** We created synthetic data sets by generating $T = 200$ task parameters $w_t$ from a 5-dimensional Gaussian distribution with zero mean and covariance equal to $\operatorname{Diag}(1, 0.25, 0.1, 0.05, 0.01)$. These are the relevant dimensions we wish to learn. To these we kept adding up to 20 irrelevant dimensions which are exactly zero. The training and test sets were selected randomly from $[0, 1]^{25}$ and contained 5 and 10 examples per task respectively. The outputs $y_{ti}$ were computed from the $w_t$ and $x_{ti}$ as $y_{ti} = \langle w_t, x_{ti} \rangle + \nu$, where $\nu$ is zero-mean Gaussian noise with standard deviation equal to 0.1.

We first present, in Figure 1, the number of features learned by our algorithm, as measured by $\operatorname{rank}(D)$. The plot on the left corresponds to a data set of 200 tasks with 25 input dimensions and that on the right to a real data set of 180 tasks described in the next subsection. As expected, the number of features decreases with $\gamma$.

Figure 2 depicts the performance of our algorithm for $T = 10, 25, 100$ and 200 tasks along with the performance of 200 independent standard ridge regressions on the data. For $T = 10, 25$ and 100, we averaged the performance metrics over runs on all the data so that our estimates have comparable variance. In agreement with past empirical and theoretical evidence (see e.g. [4]), learning multiple tasks together significantly improves on learning the tasks independently. Moreover, the performance of the algorithm improves when more tasks are available. This improvement is moderate for low dimensionalities but increases as the number of irrelevant dimensions increases.

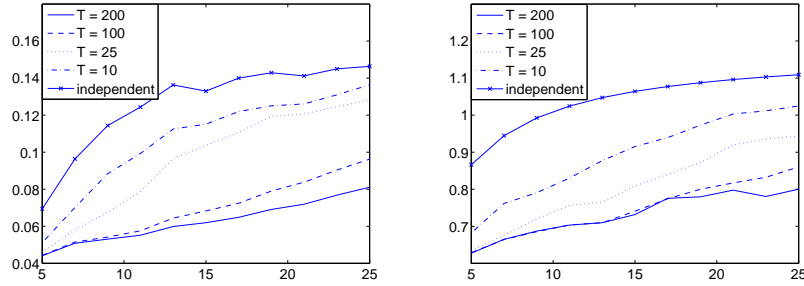

Figure 2: Test error (left) and residual of learned features (right) vs. dimensionality of the input.

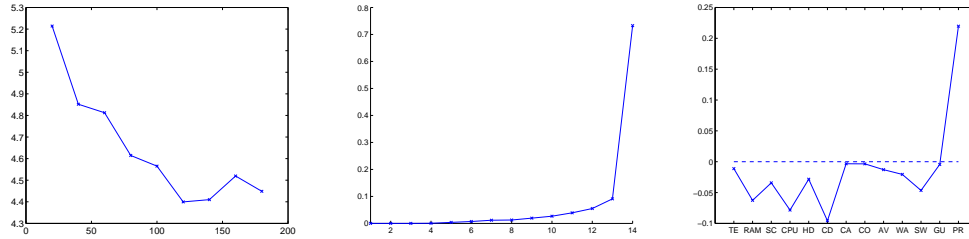

Figure 3: Test error vs. number of tasks (left) for the computer survey data set. Significance of features (middle) and attributes learned by the most important feature (right).

On the right, we have plotted a residual measure of how well the learned features approximate the actual ones used to generate the data. More specifically, we depict the Frobenius norm of the difference of the learned and actual $D$'s versus the input dimensionality. We observe that adding more tasks leads to better estimates of the underlying features.

**Conjoint analysis experiment.** We then tested the method using a real data set about people's ratings of products from [13]. The data was taken from a survey of 180 persons who rated the likelihood of purchasing one of 20 different personal computers. Here the persons correspond to tasks and the PC models to examples. The input is represented by the following 13 binary attributes: telephone hot line (TE), amount of memory (RAM), screen size (SC), CPU speed (CPU), hard disk (HD), CD-ROM/multimedia (CD), cache (CA), Color (CO), availability (AV), warranty (WA), software (SW), guarantee (GU) and price (PR). We also added an input component accounting for the bias term. The output is an integer rating on the scale $0-10$. Following [13], we used $4$ examples per task as the test data and $8$ examples per task as the training data.

As shown in Figure 3, the performance of our algorithm improves with the number of tasks. It also performs much better than independent ridge regressions, whose test error is equal to $16.53$. In this particular problem, it is also important to investigate which features are significant to all consumers and how they weight the 13 computer attributes. We demonstrate the results in the two adjacent plots, which were obtained with the data for all 180 tasks. In the middle, the distribution of the eigenvalues of $D$ is depicted, indicating that there is a single most important feature which is shared by all persons. The plot on the right shows the weight of each input dimension in this most important feature. This feature seems to weight the technical characteristics of a computer (RAM, CPU and CD-ROM) against its price. Therefore, in this application our algorithm is able to discern interesting patterns in people's decision process.

**School data.** Preliminary experiments with the school data used in [3] achieved explained variance $37.1\%$ compared to $29.5\%$ in that paper. These results will be reported in future work.

## 6 Conclusion

We have presented an algorithm which learns common sparse function representations across a pool of related tasks. To our knowledge, our approach provides the first convex optimization formulation for multi-task feature learning. Although convex optimization methods have been derived for the

simpler problem of feature selection [12], prior work on multi-task feature learning has been based on more complex optimization problems which are not convex [2, 4, 6] and, so, are at best only guaranteed to converge to a local minimum.

Our algorithm shares some similarities with recent work in [2] where they also alternately update the task parameters and the features. Two main differences are that their formulation is not convex and that, in our formulation, the number of learned features is not a parameter but it is controlled by the regularization parameter.

This work may be extended in different directions. For example, it would be interesting to explore whether our formulation can be extended to more general models for the structure across the tasks, like in [20] where ICA type features are learned, or to hierarchical feature models like in [18].

**Acknowledgments**

We wish to thank Yiming Ying and Raphael Hauser for observations on the convexity of (3.2), Charles Micchelli for valuable suggestions and the anonymous reviewers for their useful comments. This work was supported by EPSRC Grants GR/T18707/01 and EP/D071542/1, and by the IST Programme of the European Commission, under the PASCAL Network of Excellence IST-2002-506778.

## Footnotes

[1]We consider each object recognition problem within each object category, e.g. recognizing a face among faces, or a car among cars, to be a different task.

[2]As noted in the introduction, other multi-task learning methods can be used. For example, we can also penalize the variance of the $w_t$'s – "forcing" them to be close to each other – as in [8]. This would only slightly change the overall method.

# References

[1] J.Abernethy, F. Bach, T. Evgeniou and J-P. Vert. Low-rank matrix factorization with attributes. Technical report N24/06/MM, Ecole des Mines de Paris, 2006.

[2] R.K. Ando and T. Zhang. A framework for learning predictive structures from multiple tasks and unlabeled data. *J. Machine Learning Research*. 6: 1817–1853, 2005.

[3] B. Bakker and T. Heskes. Task clustering and gating for Bayesian multi–task learning. *J. of Machine Learning Research*, 4: 83–99, 2003.

[4] J. Baxter. A model for inductive bias learning. *J. of Artificial Intelligence Research*, 12: 149–198, 2000.

[5] S. Ben-David and R. Schuller. Exploiting task relatedness for multiple task learning. Proceedings of Computational Learning Theory (COLT), 2003.

[6] R. Caruana. Multi–task learning. *Machine Learning*, 28: 41–75, 1997.

[7] D. Donoho. For most large underdetermined systems of linear equations, the minimal $l^1$-norm near-solution approximates the sparsest near-solution. *Preprint*, Dept. of Statistics, Stanford University, 2004.

[8] T. Evgeniou, C.A. Micchelli and M. Pontil. Learning multiple tasks with kernel methods. *J. Machine Learning Research*, 6: 615–637, 2005.

[9] T. Evgeniou, M. Pontil and O. Toubia. A convex optimization approach to modeling consumer heterogeneity in conjoint estimation. INSEAD N 2006/62/TOM/DS.

[10] M. Fazel, H. Hindi and S. P. Boyd. A rank minimization heuristic with application to minimum order system approximation. Proceedings, American Control Conference, 6, 2001.

[11] T. Hastie, R. Tibshirani and J. Friedman. *The Elements of Statistical Learning: Data Mining, Inference and Prediction*. Springer Verlag Series in Statistics, New York, 2001.

[12] T. Jebara. Multi-task feature and kernel selection for SVMs. Proc. of ICML 2004.

[13] P.J. Lenk, W.S. DeSarbo, P.E. Green, M.R. Young. Hierarchical Bayes conjoint analysis: recovery of partworth heterogeneity from reduced experimental designs. *Marketing Science*, 15(2): 173–191, 1996.

[14] C.A. Micchelli and A. Pinkus. Variational problems arising from balancing several error criteria. Rendiconti di Matematica, Serie VII, 14: 37-86, 1994.

[15] C. A. Micchelli and M. Pontil. On learning vector–valued functions. *Neural Computation*, 17:177–204, 2005.

[16] T. Serre, M. Kouh, C. Cadieu, U. Knoblich, G. Kreiman, T. Poggio. Theory of object recognition: computations and circuits in the feedforward path of the ventral stream in primate visual cortex. AI Memo No. 2005-036, MIT, Cambridge, MA, October, 2005.

[17] N. Srebro, J.D.M. Rennie, and T.S. Jaakkola. Maximum-margin matrix factorization. NIPS 2004.

[18] A. Torralba, K. P. Murphy and W. T. Freeman. Sharing features: efficient boosting procedures for multiclass object detection. Proc. of CVPR'04, pages 762–769, 2004.

[19] K. Yu, V. Tresp and A. Schwaighofer. Learning Gaussian processes from multiple tasks. Proc. of ICML 2005.

[20] J. Zhang, Z. Ghahramani and Y. Yang. Learning Multiple Related Tasks using Latent Independent Component Analysis. NIPS 2006.
